# Practical Issues in Temporal Difference Learning

**Gerald Tesauro**
IBM Thomas J. Watson Research Center
P. O. Box 704
Yorktown Heights, NY 10598
tesauro@watson.ibm.com

## Abstract

This paper examines whether temporal difference methods for training connectionist networks, such as Suttons's TD($\lambda$) algorithm, can be successfully applied to complex real-world problems. A number of important practical issues are identified and discussed from a general theoretical perspective. These practical issues are then examined in the context of a case study in which TD($\lambda$) is applied to learning the game of backgammon from the outcome of self-play. This is apparently the first application of this algorithm to a complex nontrivial task. It is found that, with zero knowledge built in, the network is able to learn from scratch to play the entire game at a fairly strong intermediate level of performance, which is clearly better than conventional commercial programs, and which in fact surpasses comparable networks trained on a massive human expert data set. The hidden units in these network have apparently discovered useful features, a longstanding goal of computer games research. Furthermore, when a set of hand-crafted features is added to the input representation, the resulting networks reach a near-expert level of performance, and have achieved good results against world-class human play.

## 1 INTRODUCTION

We consider the prospects for applications of the TD($\lambda$) algorithm for delayed reinforcement learning, proposed in (Sutton, 1988), to complex real-world problems. TD($\lambda$) is an algorithm for adjusting the weights in a connectionist network which

has the following form:

$$\Delta w_t = \alpha(P_{t+1} - P_t) \sum_{k=1}^{t} \lambda^{t-k} \nabla_w P_k \tag{1}$$

where $P_t$ is the network's output upon observation of input pattern $x_t$ at time $t$, $w$ is the vector of weights that parameterizes the network, and $\nabla_w P_k$ is the gradient of network output with respect to weights. Equation 1 basically couples a temporal difference method for temporal credit assignment with a gradient-descent method for structural credit assigment; thus it provides a way to adapt supervised learning procedures such as back-propagation to solve temporal credit assignment problems. The $\lambda$ parameter interpolates between two limiting cases: $\lambda = 1$ corresponds to an explicit supervised pairing of each input pattern $x_t$ with the final reward signal, while $\lambda = 0$ corresponds to an explicit pairing of $x_t$ with the next prediction $P_{t+1}$.

Little theoretical guidance is available for practical uses of this algorithm. For example, one of the most important issues in applications of network learning procedures is the choice of a good representation scheme. However, the existing theoretical analysis of TD($\lambda$) applies primarily to look-up table representations in which the network has enough adjustable parameters to explicitly store the value of every possible state in the state space. This will clearly be intractable for real-world problems, and the theoretical results may be completely inappropriate, as they indicate, for example, that every possible state in the state space has to be visited infinitely many times in order to guarantee convergence.

Another important class of practical issues has to do with the nature of the task being learned, e.g., whether it is noisy or deterministic. In volatile environments with a high step-to-step variance in expected reward, TD learning is likely to be difficult. This is because the value of $P_{t+1}$, which is used as a heuristic teacher signal for $P_t$, may have nothing to do with the true value of the state $x_t$. In such cases it may be necessary to modify TD($\lambda$) by including a lookahead process which averages over the step-to-step noise.

Additional difficulties must also be expected if the task is a combined prediction-control task, in which the predictor network is used to make control decisions, as opposed to a prediction only task. As the network's predictions change, its control strategies also change, and this changes the target predictions that the network is trying to learn. In this case, theory does not say whether the combined learning system would converge at all, and if so, whether it would converge to the optimal predictor-controller. It might be possible for the system to get stuck in a self-consistent but non-optimal predictor-controller.

A final set of practical issues are algorithmic in nature, such as convergence, scaling, and the possibility of overtraining or overfitting. TD($\lambda$) has been proven to converge only for a linear network and a linearly independent set of input patterns (Sutton, 1988; Dayan, 1992). In the more general case, the algorithm may not converge even to a locally optimal solution, let alone to a globally optimal solution.

Regarding scaling, no results are available to indicate how the speed and quality of TD learning will scale with the temporal length of sequences to be learned, the dimensionality of the input space, the complexity of the task, or the size of the network. Intuitively it seems likely that the required training time might increase

dramatically with the sequence length. The training time might also scale poorly with the network or input space dimension, e.g., due to increased sensitivity to noise in the teacher signal. Another potential problem is that the quality of solution found by gradient-descent learning relative to the globally optimal solution might get progressively worse with increasing network size.

Overtraining occurs when continued training of the network results in poorer performance. Overfitting occurs when a larger network does not do as well on a task as a smaller network. In supervised learning, both of these problems are believed to be due to a limited data set. In the TD approach, training takes place on-line using patterns generated *de novo*, thus one might hope that these problems would not occur. But both overtraining and overfitting may occur if the error function minimized during training does not correspond to the performance function that the user cares about. For example, in a combined prediction-control task, the user may care only about the quality of control signals, not the absolute accuracy of the predictions.

## 2   A CASE STUDY: TD LEARNING OF BACKGAMMON STRATEGY

We have seen that existing theory provides little indication of how TD($\lambda$) will behave in practical applications. In the absence of theory, we now examine empirically the above-mentioned issues in the context of a specific application: learning to play the game of backgammon from the outcome of self-play. This application was selected because of its complexity and stochastic nature, and because a detailed comparison can be made with the alternative approach of supervised learning from human expert examples (Tesauro, 1989; Tesauro, 1990).

It seems reasonable that, by watching two fixed opponents play out a large number of games, a network could learn by TD methods to predict the expected outcome of any given board position. However, the experiments presented here study the more interesting question of whether a network can learn from its own play. The learning system is set up as follows: the network observes a sequence of board positions $x_1, x_2, ..., x_f$ leading to a final reward signal $z$. In the simplest case, $z = 1$ if White wins and $z = 0$ if Black wins. In this case the network's output $P_t$ is an estimate of White's probability of winning from board position $x_t$. The sequence of board positions is generated by setting up an initial configuration, and making plays for both sides using the network's output as an evaluation function. In other words, the move which is selected at each time step is the move which maximizes $P_t$ when White is to play and minimizes $P_t$ when Black is to play.

The representation scheme used here contained only a simple encoding of the "raw" board description (explained in detail in figure 2), and did not utilize any additional pre-computed "features" relevant to good play. Since the input encoding scheme contains no built-in knowledge about useful features, and since the network only observes its own play, we may say that this is a "knowledge-free" approach to learning backgammon. While it's not clear that this approach can make any progress beyond a random starting state, it at least provides a baseline for judging other approaches using various forms of built-in knowledge.

The approach described above is similar in spirit to Samuel's scheme for learning checkers from self-play (Samuel, 1959), but in several ways it is a more challenging learning task. Unlike the raw board description used here, Samuel's board description used a number of hand-crafted features which were designed in consultation with human checkers experts. The evaluation function learned in Samuel's study was a linear function of the input variables, whereas multilayer networks learn more complex nonlinear functions. Finally, Samuel found that it was necessary to give the learning system at least one fixed intermediate goal, material advantage, as well as the ultimate goal of the game. The proposed backgammon learning system has no such intermediate goals.

The networks had a feedforward fully-connected architecture with either no hidden units, or a single hidden layer with between 10 and 40 hidden units. The learning algorithm parameters were set, after a certain amount of parameter tuning, at $\alpha = 0.1$ and $\lambda = 0.7$.

The average sequence length appeared to depend strongly on the quality of play. With decent play on both sides, the average game length is about 50-60 time steps, whereas for the random initial networks, games often last several hundred or even several thousand time steps. This is one of the reasons why the proposed self-learning scheme appeared unlikely to work.

Learning was assessed primarily by testing the networks in actual game play against Sun Microsystems' Gammontool program. Gammontool is representative of the playing ability of typical commercial programs, and provides a decent benchmark for measuring game-playing strength: human beginners can win about 40% of the time against it, decent intermediate-level humans would win about 60%, and human experts would win about 75%. (The random initial networks before training win only about 1%.)

Networks were trained on the entire game, starting from the opening position and going all the way to the end. This is an admittedly naive approach which was not expected to yield any useful results other than a reference point for judging more sensible approaches. However, the rather surprising result was that a significant amount of learning actually took place. Results are shown in figure 1. For comparison purposes, networks with the same input coding scheme were also trained on a massive human expert data base of over 15,000 engaged positions, following the training procedure described in (Tesauro, 1989). These networks were also tested in game play against Gammontool.

Given the complexity of the task, size of input space and length of typical sequences, it seems remarkable that the TD nets can learn on their own to play at a level substantially better than Gammontool. Perhaps even more remarkable is that the TD nets surpass the EP nets trained on a massive human expert data base: the best TD net won 66.2% against Gammontool, whereas the best EP net could only manage 59.4%. This was confirmed in a head-to-head test in which the best TD net played 10,000 games against the best EP net. The result was 55% to 45% in favor of the TD net. This confirms that the Gammontool benchmark gives a reasonably accurate measure of relative game-playing strength, and that the TD net really is better than the EP net. In fact, the TD net with no features appears to be as good as Neurogammon 1.0, backgammon champion of the 1989 Computer

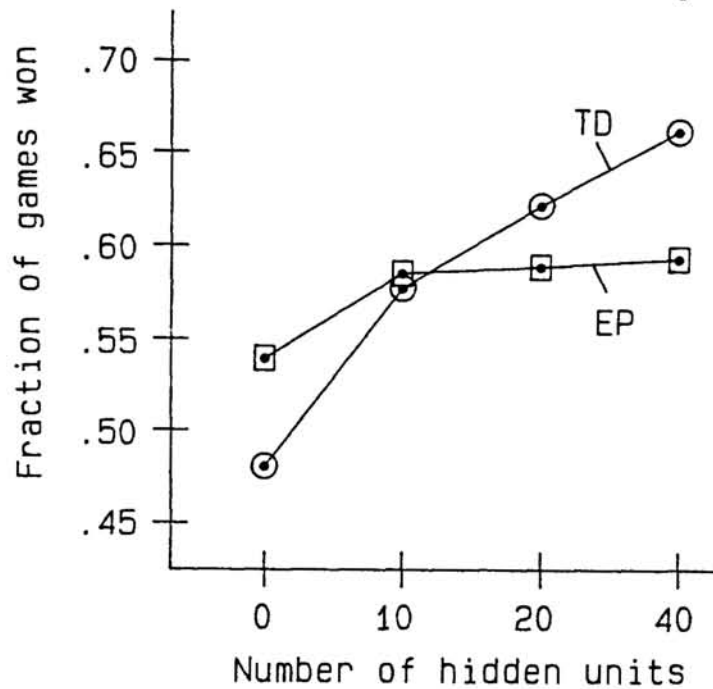

Figure 1: Plot of game performance against Gammontool vs. number of hidden units for networks trained using TD learning from self-play (TD), and supervised training on human expert preferences (EP). Each data point represents the result of a 10,000 game test, and should be accurate to within one percentage point.

Olympiad, which does have features, and which wins 65% against Gammontool. A 10,000 game test of the best TD net against Neurogammon 1.0 yielded statistical equality: 50% for the TD net and 50% for Neurogammon.

It is also of interest to examine the weights learned by the TD nets, shown in figure 2. One can see a great deal of spatially organized structure in the pattern of weights, and some of this structure can be interpreted as useful features by a knowledgable backgammon player. For example, the first hidden unit in figure 2 appears to be a race-oriented feature detector, while the second hidden unit appears to be an attack-oriented feature detector. The TD net has apparently solved the long-standing "feature discovery" problem, which was recently stated in (Frey, 1986) as follows: "Samuel was disappointed in his inability to develop a mechanical strategy for defining features. He thought that true machine learning should include the discovery and definition of features. Unfortunately, no one has found a practical way to do this even though more than two and a half decades have passed."

The training times needed to reach the levels of performance shown in figure 1 were on the order of 50,000 training games for the networks with 0 and 10 hidden units, 100,000 games for the 20-hidden unit net, and 200,000 games for the 40-hidden unit net. Since the number of training games appears to scale roughly linearly with the number of weights in the network, and the CPU simulation time per game on a serial computer also scales linearly with the number of weights, the total CPU time thus scales quadratically with the number of weights: on an IBM RS/6000 workstation, the smallest network was trained in several hours, while the largest net required two weeks of simulation time.

In qualitative terms, the TD nets have developed a style of play emphasizing run-

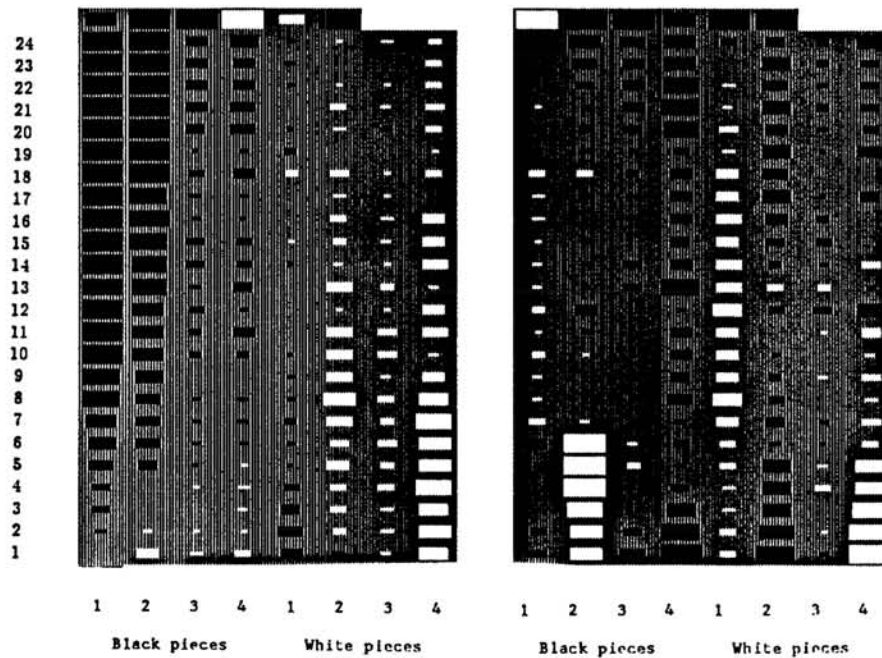

Figure 2: Weights from the input units to two hidden units in the best TD net. Black squares represent negative weights; white squares represent positive weights; size indicates magnitude of weights. Rows represent spatial locations 1-24, top row represents no. of barmen, men off, and side to move. Columns represent number of Black and White men as indicated. The first hidden unit has two noteworthy features: a linearly increasing pattern of negative weights for Black blots and Black points, and a negative weighting of White men off and a positive weighting of Black men off. These contribute to an estimate of Black's probability of winning based on his racing lead. The second hidden unit has the following noteworthy features: strong positive weights for Black home board points, strong positive weights for White men on bar, positive weights for White blots, and negative weights for White points in Black's home board. These factors all contribute to the probability of a successful Black attacking strategy.

ning and tactical play, whereas the EP nets favor more quiescent positional play emphasizing blocking rather than racing. This is more in line with human expert play, but it often leads to complex prime vs. prime and back-game situations that are hard for the network to evaluate properly. This suggests one possible advantage of the TD approach over the EP approach: by imitating an expert teacher, the learner may get itself into situations that it can't handle. With the alternative approach of learning from experience, the learner may not reproduce the expert strategies, but at least it will learn to handle whatever situations are brought about by its own strategy.

It's also interesting that TD net plays well in early phases of play, whereas its play becomes worse in the late phases of the game. This is contrary to the intuitive notion that states far from the end of the sequence should be harder to learn than states near the end. Apparently the inductive bias due to the representation scheme and network architecture is more important than temporal distance to the final outcome.

# 3   TD LEARNING WITH BUILT-IN FEATURES

We have seen that TD networks with no built-in knowledge are able to reach computer championship levels of performance for this particular application. It is then natural to wonder whether even greater levels of performance might be obtained by adding hand-crafted features to the input representation. In a separate series of experiments, TD nets containing all of Neurogammon's features were trained from self-play as described in the previous section. Once again it was found that the performance improved monotonically by adding more hidden units to the network, and training for longer training times. The best performance was obtained with a network containing 80 hidden units and over 25,000 weights. This network was trained for over 300,000 training games, taking over a month of CPU time on an RS/6000 workstation. The resulting level of performance was 73% against Gammontool and nearly 60% against Neurogammon. This is very close to a human expert level of performance, and is the strongest program ever seen by this author.

The level of play of this network was also tested in an all-day match against two-time World Champion Bill Robertie, one of the world's best backgammon players. At the end of the match, a total of 31 games had been played, of which Robertie won 18 and the TD net 13. This showed that the TD net was capable of a respectable showing against world-class human play. In fact, Robertie thinks that the network's level of play is equal to the average good human tournament player.

It's interesting to speculate about how far this approach can be carried. Further substantial improvements might be obtained by training much larger networks on a supercomputer or special-purpose hardware. On such a machine one could also search beyond one ply, and there is some evidence that small-to-moderate improvements could be obtained by running the network in two-ply search mode. Finally, the features in Berliner's BKG program (Berliner, 1980) or in some of the top commercial programs are probably more sophisticated than Neurogammon's relatively simple features, and hence might give better performance. The combination of all three improvements (bigger nets, two-ply search, better features) could conceivably result in a network capable of playing at world-class level.

# 4   CONCLUSIONS

The experiments in this paper were designed to test whether TD($\lambda$) could be successfully applied to complex, stochastic, nonlinear, real-world prediction and control problems. This cannot be addressed within current theory because it cannot answer such basic questions as whether the algorithm converges or how it would scale.

Given the lack of any theoretical guarantees, the results of these experiments are very encouraging. Empirically the algorithm always converges to at least a local minimum, and the quality of solution generally improves with increasing network size. Furthermore, the scaling of training time with the length of input sequences, and with the size and complexity of the task, does not appear to be a serious problem. This was ascertained through studies of simplified endgame situations, which took about as many training games to learn as the full-game situation (Tesauro, 1992). Finally, the network's move selection ability is better than one would expect based on its prediction accuracy. The absolute prediction accuracy is only at

the 10% level, whereas the difference in expected outcome between optimal and non-optimal moves is usually at the level of 1% or less.

The most encouraging finding, however, is a clear demonstration that TD nets with zero built-in knowledge can outperform identical networks trained on a massive data base of expert examples. It would be nice to understand exactly how this is possible. The ability of TD nets to discover features on their own may also be of some general importance in computer games research, and thus worthy of further analysis.

Beyond this particular application area, however, the larger and more important issue is whether learning from experience can be useful and practical for more general complex problems. The quality of results obtained in this study indicates that the approach may work well in practice. There may also be some intrinsic advantages over supervised training on a fixed data set. At the very least, for tasks in which the exemplars are hand-labeled by humans, it eliminates the laborious and time-consuming process of labeling the data. Furthermore, the learning system would not be fundamentally limited by the quantity of labeled data, or by errors in the labeling process. Finally, preserving the intrinsic temporal nature of the task, and informing the system of the consequences of its actions, may convey important information about the task which is not necessarily contained in the labeled exemplars. More theoretical and empirical work will be needed to establish the relative advantages and disadvantages of the two approaches; this could result in the development of hybrid algorithms combining the best of both approaches.

## References

H. Berliner, "Computer backgammon." *Sci. Am.* **243**:1, 64-72 (1980).

P. Dayan, "Temporal differences: TD($\lambda$) for general $\lambda$." *Machine Learning*, in press (1992).

P. W. Frey, "Algorithmic strategies for improving the performance of game playing programs." In: D. Farmer et al. (Eds.), *Evolution, Games and Learning*. Amsterdam: North Holland (1986).

A. Samuel, "Some studies in machine learning using the game of checkers." *IBM J. of Research and Development* **3**, 210-229 (1959).

R. S. Sutton, "Learning to predict by the methods of temporal differences." *Machine Learning* **3**, 9-44 (1988).

G. Tesauro and T. J. Sejnowski, "A parallel network that learns to play backgammon." *Artificial Intelligence* **39**, 357-390 (1989).

G. Tesauro, "Connectionist learning of expert preferences by comparison training." In D. Touretzky (Ed.), *Advances in Neural Information Processing* **1**, 99-106 (1989).

G. Tesauro, "Neurogammon: a neural network backgammon program." *IJCNN Proceedings* III, 33-39 (1990).

G. Tesauro, "Practical issues in temporal difference learning." *Machine Learning*, in press (1992).
